# Image Recognition in Context: Application to Microscopic Urinalysis

**Xubo Song***
Department of Electrical and Computer Engineering
Oregon Graduate Institute of Science and Technology
Beaverton, OR 97006
*xubosong@ece.ogi.edu*

**Joseph Sill**
Department of Computation and Neural Systems
California Institute of Technology
Pasadena, CA 91125
*joe@busy.work.caltech.edu*

**Yaser Abu-Mostafa**
Department of Electrical Engineering
California Institute of Technology
Pasadena, CA 91125
*yaser@work.caltech.edu*

**Harvey Kasdan**
International Remote Imaging Systems, Inc.
Chatsworth, CA 91311

## Abstract

We propose a new and efficient technique for incorporating contextual information into object classification. Most of the current techniques face the problem of exponential computation cost. In this paper, we propose a new general framework that incorporates *partial* context at a linear cost. This technique is applied to microscopic urinalysis image recognition, resulting in a significant improvement of recognition rate over the context free approach. This gain would have been impossible using conventional context incorporation techniques.

## 1 BACKGROUND: RECOGNITION IN CONTEXT

There are a number of pattern recognition problem domains where the classification of an object should be based on more than simply the appearance of the object itself. In remote sensing image classification, where each pixel is part of ground cover, a pixel is more likely to be a glacier if it is in a mountainous area, than if surrounded by pixels of residential areas. In text analysis, one can expect to find certain letters occurring regularly in particular arrangement with other letters(qu, ee,est, tion, *etc.*). The information conveyed by the accompanying entities is referred to as *contextual information*. Human experts apply contextual information in their decision making [2][6]. It makes sense to design techniques and algorithms to make computers aggregate and utilize a more complete set of information in their decision making the way human experts do. In pattern recognition systems, however,

the primary (and often only) source of information used to identify an object is the set of measurements, or *features*, associated with the object itself. Augmenting this information by incorporating context into the classification process can yield significant benefits.

Consider a set of $N$ objects $T_i$, $i = 1, ... N$. With each object we associate a class label $c_i$ that is a member of a label set $\Omega = \{1, ..., D\}$. Each object $T_i$ is characterized by a set of measurements $\mathbf{x}_i \in \mathbf{R}^P$, which we call a feature vector. Many techniques [1][2][4][6] incorporate context by conditioning the posterior probability of objects' identities on the joint features of all accompanying objects, *i.e.*, $p(c_1, c_2, ..., c_N | \mathbf{x}_1, ..., \mathbf{x}_N)$, and then maximizing it with respect to $c_1, c_2, ..., c_N$. It can be shown that $p(c_1, c_2, ..., c_N | \mathbf{x}_1, ..., \mathbf{x}_N) \propto p(c_1 | \mathbf{x}_1) ... p(c_N | \mathbf{x}_N) \frac{p(c_1, ..., c_N)}{p(c_1) ... p(c_N)}$ given certain reasonable assumptions.

Once the context-free posterior probabilities $p(c_i | x_i)$ are known, e.g. through the use of a standard machine learning model such as a neural network, computing $p(c_1, ..., c_N | \mathbf{x}_1, ..., \mathbf{x}_N)$ for all possible $c_1, ..., c_N$ would entail $(2N + 1)D^N$ multiplications, and finding the maximum has complexity of $D^N$, which is intractable for large $N$ and $D$. [2]

Another problem with this formulation is the estimation of the high dimensional joint distribution $p(c_1, ..., c_N)$, which is ill-posed and data hungry.

One way of dealing with these problems is to limit context to local regions. With this approach, only the pixels in a close neighborhood, or letters immediately adjacent are considered [4][6][7]. Such techniques may be ignoring useful information, and will not apply to situations where context doesn't have such locality, as in the case of microscopic urinalysis image recognition. Another way is to simplify the problem using specific domain knowledge [1], but this is only possible in certain domains.

These difficulties motivate the efficient incorporation of *partial context* as a general framework, formulated in section 2. In section 3, we discuss microscopic urinalysis image recognition, and address the importance of using context for this application. Also in section 3, techniques are proposed to identify relevant context. Empirical results are shown in section 4, followed by discussions in section 5.

## 2   FORMULATION FOR INCORPORATION OF PARTIAL CONTEXT

To avoid the exponential computational cost of using the identities of all accompanying objects directly as context, we use "partial context", denoted by $A$. It is called "partial" because it is *derived* from the class labels, as opposed to consisting of an explicit labelling of all objects. The physical definition of $A$ depends on the problem at hand. In our application, $A$ represents the presence or absence of certain classes. Then the posterior probability of an object $T_i$ having class label $c_i$ conditioned on its feature vector and the relevant context $A$ is

$$p(c_i | x_i, A) = \frac{p(c_i, x_i; A)}{p(x_i; A)} = \frac{p(x_i | c_i, A) P(c_i; A)}{p(x_i; A)}$$

We assume that the feature distribution of an object depends only on its own class, *i.e.*, $p(x_i | c_i, A) = p(x_i | c_i)$. This assumption is roughly true for most real world problems. Then,

$$p(c_i|x_i, A) = \frac{p(x_i|c_i)p(c_i; A)}{p(x_i; A)} = p(c_i|x_i)\frac{p(c_i|A)}{p(c_i)}\frac{p(A)p(x_i)}{p(x_i; A)}$$
$$\propto p(c_i|x_i)\frac{p(c_i|A)}{p(c_i)} = p(c_i|x_i)\rho(c_i, A)$$

where $\rho(c_i, A) = \frac{p(c_i|A)}{p(c_i)}$ is called the *context ratio*, through which context plays its role. The context-sensitive posterior probability $p(c_i|x_i, A)$ is obtained through the context-free posterior probability $p(c_i|x_i)$ modified by the context ratio $\rho(c_i, A)$.

The partial-context maximum likelihood decision rule chooses class label $\hat{c}_i$ for element $i$ such that

$$\hat{c}_i = \underset{c_i}{\text{argmax}}\, p(c_i|\mathbf{x}_i, A) \tag{1}$$

A systematic approach to identify relevant context $A$ is addressed in section 3.3.

The partial-context approach treats each element in a set individually, but with additional information from the context-bearing factor $A$. Once $p(c_i|x_i)$ are known for all $i = 1, \ldots, N$, and the context $A$ is obtained, to maximize $p(c_i|x_i, A)$ from $D$ possible values that $c_i$ can take on and for all $i$, the total number of multiplications is $2N$, and the complexity for finding the maximum is $ND$. Both are linear in $N$. The density estimation part is also trivial since it is very easy to estimate $p(c|A)$.

## 3 MICROSCOPIC URINALYSIS

### 3.1 INTRODUCTION

Urine is one of the most complex body fluid specimens: it potentially contains about 60 meaningful types of elements. Microscopic urinalysis detects the presence of elements that often provide early diagnostic information concerning dysfunction, infection, or inflammation of the kidneys and urinary tract. Thus this non-invasive technique can be of great value in clinical case management. Traditional manual microscopic analysis relies on human operators who read the samples visually and identify them, and therefore is time-consuming, labor-intensive and difficult to standardize. Automated microscopy of all specimens is more practical than manual microscopy, because it eliminates variation among different technologists. This variation becomes more pronounced when the same technologist examines increasing numbers of specimens. Also, it is less labor-intensive and thus less costly than manual microscopy. It also provides more consistent and accurate results. An automated urinalysis system workstation (The $YellowIRIS^{TM}$, International Remote Imaging Systems, Inc.) has been introduced in numerous clinical laboratories for automated microscopy. Urine samples are processed and examined at 100x (low power field) and 400x magnifications (high power field) with bright-field illumination. The $YellowIRIS^{TM}$ automated system collects video images of formed analytes in a stream of uncentrifuged urine passing an optical assembly. Each image has one analyte in it. These images are given to a computer algorithm for automatic identification of analytes.

Context is rich in urinalysis and plays a crucial role in analyte classification. Some combinations of analytes are more likely than others. For instance, the presence of bacteria indicates the presence of white blood cells, since bacteria tend to cause infection and thus trigger the production of more white blood cells. If amorphous crystals show up, they tend to show up in bunches and in all sizes. Therefore, if there are amorphous crystal look-alikes in various sizes, it is quite possible that they are amorphous crystals. Squamous epithelial cells can appear both flat or rolled up. If squamous epithelial cells in one form are detected,

Table 1: Features extracted from urine anylates images

| feature number | feature description |
|---|---|
| 1 | area |
| 2 | length of edge |
| 3 | $\frac{\text{square root of area}}{\text{length of edge}}$ |
| 4 | $\frac{\text{standard deviation}}{\text{mean}}$ of distance from center to edge |
| 5 | $\frac{\lambda_1}{\lambda_2}$ |
| 6 | $\frac{\text{sum of length of two longest straight edges}}{\text{total length of edge}}$ |
| 7 | $\frac{\text{sum of length of four longest straight edges}}{\text{total length of edge}}$ |
| 8 | $\frac{\text{sum of length of two longest semi-straight edges}}{\text{total length of edge}}$ |
| 9 | $\frac{\text{sum of length of four longest semi-straight edges}}{\text{total length of edge}}$ |
| 10 | the mean of red distribution |
| 11 | the mean of blue distribution |
| 12 | the mean of green distribution |
| 13 | $15^{th}$ percentile of gray level histogram |
| 14 | $85^{th}$ percentile of gray level histogram |
| 15 | the standard deviation of gray level intensity |
| 16 | energy of the Laplacian transformation of grey level image |

then it is likely that there are squamous epithelial cells in the other form. Utilizing such context is crucial for classification accuracy.

The classes we are looking at are bacteria, calcium oxalate crystals, red blood cells, white blood cells, budding yeast, amorphous crystals, uric acid crystals, and artifacts. The task of automated microscopic urinalysis is, given a urine specimen that consists of up to a few hundred images of analytes, to classify each analyte into one of these classes. The automated urinalysis system we developed consists of three steps: image processing and feature extraction, learning and pattern recognition, and context incorporation. Figure 1 shows some example analyte images. Table 1 gives a list of features extracted from analyte images.[1]

## 3.2   CONTEXT-FREE CLASSIFICATION

The features are fed into a nonlinear feed-forward neural network with 16 inputs, 15 hidden units with sigmoid transfer functions, and 8 sigmoid output units. A cross-entropy error function is used in order to give the output a probability interpretation. Denote the input feature vector as $\mathbf{x}$, the network outputs a $D$ dimensional vector ($D = 8$ in our case) $\mathbf{p} = \{p(d|\mathbf{x})\}, d = 1, ..., D$, where $p(d|\mathbf{x})$ is

$$p(d|\mathbf{x}) = Prob(\text{ an analyte belongs to class } d | \text{ feature } \mathbf{x})$$

The decision made at this stage is

$$d(\mathbf{x}) = \underset{d}{\operatorname{argmax}} \ p(d|\mathbf{x})$$

## 3.3   IDENTIFICATION OF RELEVANT PARTIAL CONTEXT

Not all classes are relevant in terms of carrying contextual information. We propose three criteria based on which we can systematically investigate the relevance of the class presence. To use these criteria, we need to know the following distributions: the class prior distribution $p(c)$ for $c = 1, \ldots, D$; the conditional class distribution $p(c|A_d)$ for $c = 1, \ldots, D$

and $d = 1, \ldots, D$; and the class presence prior distribution $p(A_d)$ for $d = 1, \ldots, D$. $A_d$ is a binary random variable indicating the presence of class $d$. $A_d = 1$ if class $d$ is present, and $A_d = 0$ otherwise. All these distributions can be easily estimated from the database.

The first criterion is the correlation coefficient between the presence of any two classes; the second one is the classical mutual information $I(c; A_d)$ between the presence of a class $A_d$ and the class probability $p(c)$, where $I(c; A_d)$ is defined as $I(c; A_d) = H(c) - H(c|A_d)$ where $H(c) = \sum_{i=1}^{D} p(c = i) ln(p(c = i))$ is the entropy of the class priors and $H(c|A_d) = P(A_d = 1)H(c|A_d = 1) + P(A_d = 0)H(c|A_d = 0)$ is the conditional entropy of $c$ conditioned on $A_d$. The third criterion is what we call the *expected relative entropy* $D(c||A_d)$ between the presence of a class $A_d$ and the labeling probability $p(c)$, which we define as $D(c||A_d) = P(A_d = 1)D(p(c)||p(c|A_d = 1)) + P(A_d = 0)D(p(c)||p(c|A_d = 0))$ where $D(p(c)||p(c|A_d = 1)) = \sum_{i=1}^{D} p(c = i|A_d = 1) ln(\frac{p(c=i|A_d=1)}{p(c=i)})$ and $D(p(c)||p(c|A_d = 0)) = \sum_{i=1}^{D} p(c = i|A_d = 0) ln(\frac{p(c=i|A_d=0)}{p(c=i)})$

According to the first criterion, one type of analyte is considered relevant to another if the absolute value of their correlation coefficient is beyond a certain threshold. It shows that uric acid crystals, budding yeast and calcium oxalate crystals are not relevant to any other types even by a generous threshold of 0.10. Similarly, the bigger the mutual information between the presence of a class and the class distribution, the more relevant this class is. Ranking the analyte types in terms of $I(c; A_d)$ in a descending manner gives rise to the following list: bacteria, amorphous crystals, red blood cells, white blood cells, uric acid crystals, budding yeast and calcium oxalate crystals. Once again, ranking the analyte types in terms of $D(c||A_d)$ in a descending manner gives rise to the following list: bacteria, red blood cells, amorphous crystals, white blood cells, calcium oxalate crystals, budding yeast and uric acid crystals. All three criteria lead to similar conclusions regarding the relevance of class presence – bacteria, red blood cells, amorphous crystals, and white blood cells are relevant, while calcium oxalate crystals, budding yeast and uric acid crystals are not. (Baed on prior knowledge, we discard artifacts from the outset as an irrelevant class.)

### 3.4 ALGORITHM FOR INCORPORATING PARTIAL CONTEXT

Once the $M$ relevant classes are identified, the following algorithm is used to incorporate partial context.

**Step 0** Estimate the priors $p(c|A_d)$ and $p(c)$, for $c \in \{1, 2, \ldots, D\}$ and $d \in \{1, 2, \ldots, D\}$.

**Step 1** For a given $x_i$, compute $p(c_i|x_i)$ for $c_i = 1, 2, \ldots, D$ using whichever base machine learning model is preferred ( in our case, a neural network).

**Step 2** Let the $M$ relevant classes be $R_1, \ldots, R_M$. According to the no-context $p(c_i|x_i)$ and certain criteria for detecting the presence or absence of all the relevant classes, get $A_{R_1}, \ldots, A_{R_M}$.

**Step 3** Let $p(c_i|x_i, \underline{A_0}) = p(c_i|x_i)$, where $A_0$ is the null element. Incorporate context from each relevant class sequentially, *i.e.*, for $m = 1$ to $M$, iteratively compute

$$p(c_i|x_i; \underline{A_0, \ldots, A_{R_{m-1}}, A_{R_m}}) = p(c_i|x_i, \underline{A_0, \ldots, A_{R_{m-1}}}) \frac{p(c_i|A_{R_m})p(A_{R_m})}{p(c)}$$

**Step 4** Recompute $A_{R_1}, \ldots, A_{R_M}$ based on the new class labellings. Return to step 3 and repeat until algorithm converges.[2]

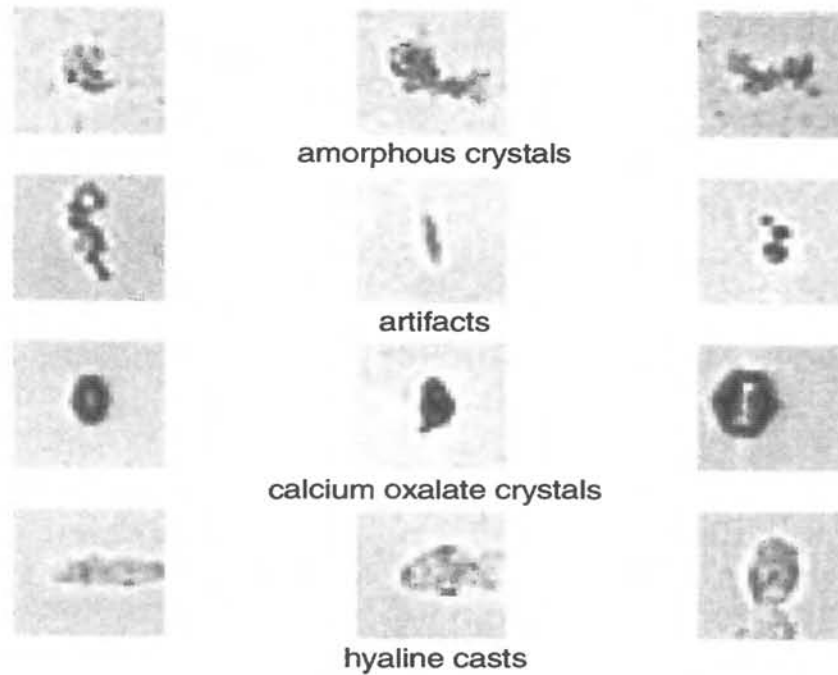

amorphous crystals

artifacts

calcium oxalate crystals

hyaline casts

Figure 1: Example of some of the analyte images.

**Step 5** Label the objects according to the final context-containing $p(c_i|\mathbf{x}_i, A_{R_1}, \ldots, A_{R_M})$, *i.e.*, $\hat{c}_i = \underset{c_i}{\operatorname{argmax}}\, p(c_i|\mathbf{x}_i, A_{R_1}, \ldots, A_{R_M})$ for $i = 1, \ldots, N$.

This algorithm is invariant with respect to the ordering of the $M$ relevant classes in $(A_1, \ldots, A_M)$. The proof is omitted here.

## 4   RESULTS

The algorithm using partial context was tested on a database of 83 urine specimens, containing a total of 20,276 analyte images. Four classes are considered relevant according to the criteria described in section 3.3: bacteria, red blood cells, white blood cells and amorphous crystals. We measure two types of error: analyte-by-analyte error, and specimen diagnostic error. The average analyte-by-analyte error is reduced from 44.48% before using context to 36.66% after, resulting a relative error reduction of 17.6% (Table 2). The diagnosis for a specimen is either normal or abnormal. Tables 3 and 4 compare the diagnostic performance with and without using context, and Table 5 lists the relative changes. We can see using context significantly increases correct diagnosis for both normal and abnormal specimens, and reduces both false positives and false negatives.

|  | without context | with context |
|---|---|---|
| average element-by-element error | 44.48 % | 36.66 % |

Table 2: Comparison of using and not using contextual information for analyte-by-analyte error.

probable class labels given the context and determining the context given the class labels.

|  | estimated normal | estimated abnormal |
|---|---|---|
| truly normal | 40.96 % | 7.23 % |
| truly abnormal | 19.28 % | 32.53 % |

Table 3: Diagnostic confusion matrix not using context

|  | estimated normal | estimated abnormal |
|---|---|---|
| truly normal | 42.17 % | 6.02 % |
| truly abnormal | 16.87 % | 34.94 % |

Table 4: Diagnostic confusion matrix using context

|  | estimated normal | estimated abnormal |
|---|---|---|
| truly normal | + 2.95 % | -16.73 % |
| truly abnormal | - 12.50 % | +7.41 % |

Table 5: Relative accuracy improvement (diagonal elements) and error reduction (off diagonal elements) in the diagnostic confusion matrix by using context.

## 5  CONCLUSIONS

We proposed a novel framework that can incorporate context in a simple and efficient manner, avoiding exponential computation and high dimensional density estimation. The application of the *partial context* technique to microscopic urinalysis image recognition demonstrated the efficacy of the algorithm. This algorithm is not domain dependent, thus can be readily generalized to other pattern recognition areas.

## ACKNOWLEDGEMENTS

The authors would like to thank Alexander Nicholson, Malik Magdon-Ismail, Amir Atiya at the Caltech Learning Systems Group for helpful discussions.

## Footnotes

*Author for correspondence

[1] $\lambda_1$ and $\lambda_2$ are respectively the larger and the smaller eigenvalues of the second moment matrix of an image.

[2] Hence, the algorithm has an E-M flavor, in that it goes back and forth between finding the most

## References

[1] Song, X.B. & Sill, J. & Abu-Mostafa & Harvey Kasdan, (1997) "Incorporating Contextual Information in White Blood Cell Identification", In M. Jordan, M.J. Kearns and S.A. Solla (eds.), *Advances in Neural Information Processing Systems 7*, 1997, pp. 950-956. Cambridge, MA: MIT Press.

[2] Song, Xubo (1999) "Contextual Pattern Recognition with Application to Biomedical Image Identification", Ph.D. Thesis, California Institute of Science and Technology.

[3] Boehringer-Mannheim-Corporation, Urinalysis Today, Boehringer-Mannheim-Corporation, 1991.

[4] Kittler, J.."Relaxation labelling", Pattern Recognition Theory and Applications, 1987, pp. 99-108., Pierre A. Devijver and Josef Kittler, Editors, Springer-Verlag.

[5] Kittler, J. & Illingworth, J., "Relaxation Labelling Algorithms - A Review", Image and Vision Computing, 1985, vol. 3, pp. 206-216.

[6] Toussaint, G., "The Use of Context in Pattern Recognition", Pattern Recognition, 1978, vol. 10, pp. 189-204.

[7] Swain, P. & Vardeman, S. & Tilton, J., "Contextual Classification of Multispectral Image Data", Pattern Recognition, 1981, Vol. 13, No. 6, pp. 429-441.